# Automatic Learning Rate Maximization by On-Line Estimation of the Hessian's Eigenvectors

**Yann LeCun,[1] Patrice Y. Simard,[1] and Barak Pearlmutter[2]**
[1]AT&T Bell Laboratories 101 Crawfords Corner Rd, Holmdel, NJ 07733
[2]CS&E Dept. Oregon Grad. Inst., 19600 NW vonNeumann Dr, Beaverton, OR 97006

## Abstract

We propose a very simple, and well principled way of computing the optimal step size in gradient descent algorithms. The on-line version is very efficient computationally, and is applicable to large backpropagation networks trained on large data sets. The main ingredient is a technique for estimating the principal eigenvalue(s) and eigenvector(s) of the objective function's second derivative matrix (Hessian), which does not require to even calculate the Hessian. Several other applications of this technique are proposed for speeding up learning, or for eliminating useless parameters.

## 1  INTRODUCTION

Choosing the appropriate learning rate, or step size, in a gradient descent procedure such as backpropagation, is simultaneously one of the most crucial and expert-intensive part of neural-network learning. We propose a method for computing the best step size which is both well-principled, simple, very cheap computationally, and, most of all, applicable to on-line training with large networks and data sets. Learning algorithms that use Gradient Descent minimize an objective function $E$ of the form

$$E(W) = \frac{1}{P}\sum_{p=0}^{P} E^p(W) \qquad E^p = E(W, X^p) \qquad (1)$$

where $W$ is the vector of parameters (weights), $P$ is the number of training patterns, and $X^p$ is the $p$-th training example (including the desired output if necessary). Two basic versions of gradient descent can be used to minimize $E$. In the first version, called the *batch* version, the exact gradient of $E$ with respect to $W$ is calculated, and the weights are updated by iterating the procedure

$$W \leftarrow W - \eta \nabla E(W) \qquad (2)$$

where $\eta$ is the *learning rate* or step size, and $\nabla E(W)$ is the gradient of $E$ with respect to $W$. In the second version, called *on-line*, or *Stochastic* Gradient Descent, the weights are updated after each pattern presentation

$$W \leftarrow W - \eta \nabla E^p(W) \qquad (3)$$

Before going any further, we should emphasize that our main interest is in training large networks on large data sets. As many authors have shown, Stochastic Gradient Descent (SGD) is much faster on large problems than the "batch" version. In fact, on large problems, a carefully tuned SGD algorithm outperforms most accelerated or second-order batch techniques, including Conjugate Gradient. Although there have been attempts to "stochasticize" second-order algorithms (Becker and Le Cun, 1988) (Moller, 1992), most of the resulting procedures also rely on a global scaling parameter similar to $\eta$. Therefore, there is considerable interest in finding ways of optimizing $\eta$.

## 2   COMPUTING THE OPTIMAL LEARNING RATE: THE RECIPE

In a somewhat unconventional way, we first give our simple "recipe" for computing the optimal learning rate $\eta$. In the subsequent sections, we sketch the theory behind the recipe.

Here is the proposed procedure for estimating the optimal learning rate in a back-propagation network trained with Stochastic Gradient Descent. Equivalent procedures for other adaptive machines are straightforward. In the following, the notation $\mathcal{N}(V)$ designates the normalized vector $V/||V||$. Let $W$ be the $N$ dimensional weight vector,

1. pick a normalized, $N$ dimensional vector $\Psi$ at random. Pick two small positive constants $\alpha$ and $\gamma$, say $\alpha = 0.01$ and $\gamma = 0.01$.

2. pick a training example (input and desired output) $X^p$. Perform a regular forward prop and a backward prop. Store the resulting gradient vector $G_1 = \nabla E^p(W)$.

3. add $\alpha \mathcal{N}(\Psi)$ to the current weight vector $W$,

4. perform a forward prop and a backward prop on the same pattern using the perturbed weight vector. Store the resulting gradient vector $G_2 = \nabla E^p(W + \alpha \mathcal{N}(\Psi))$

5. update vector $\Psi$ with the running average formula $\Psi \leftarrow (1 - \gamma)\Psi + \frac{\gamma}{\alpha}(G_2 - G_1)$.

6. restore the weight vector to its original value $W$.

7. loop to step 2 until $||\Psi||$ stabilizes.

8. set the learning rate $\eta$ to $||\Psi||^{-1}$, and go on to a regular training session.

The constant $\alpha$ controls the size of the perturbation. A small $\alpha$ gives a better estimate, but is more likely to cause numerical errors. $\gamma$ controls the tradeoff between the convergence speed of $\Psi$ and the accuracy of the result. It is better to start with

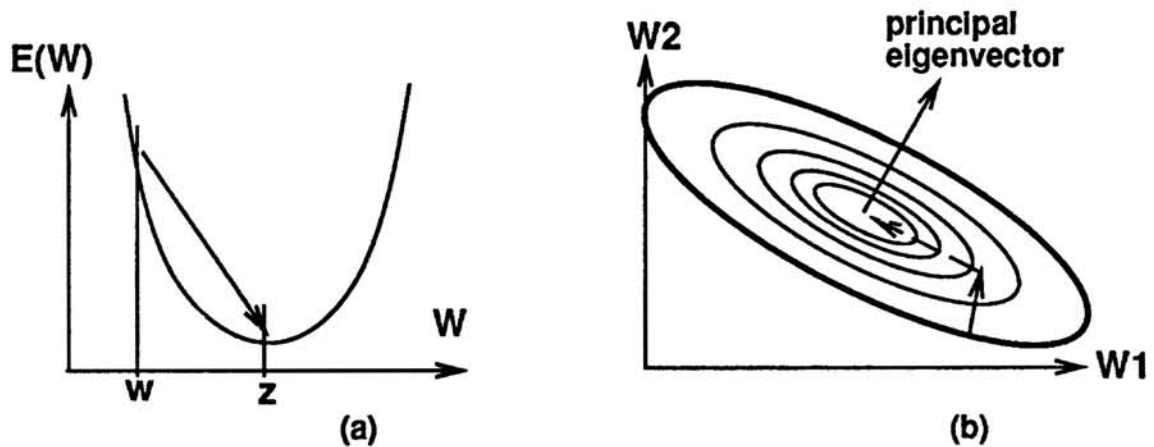

Figure 1: Gradient descent with optimal learning rate in (a) one dimension, and (b) two dimensions (contour plot).

a relatively large $\gamma$ (say 0.1) and progressively decrease it until the fluctuations on $\|\Psi\|$ are less than say 10%. In our experience accurate estimates can be obtained with between one hundred and a few hundred pattern presentations: for a large problem, the cost is very small compared to a single learning epoch.

## 3    STEP SIZE, CURVATURE AND EIGENVALUES

The procedure described in the previous section makes $\|\Psi\|$ converge to the largest positive eigenvalue of the second derivative matrix of the average objective function. In this section we informally explain why the best learning rate is the inverse of this eigenvalue. More detailed analysis of gradient descent procedures can be found in Optimization, Statistical Estimation, or Adaptive Filtering textbooks (see for example (Widrow and Stearns, 1985)). For didactical purposes, consider an objective function of the form $E(w) = \frac{h}{2}(w - z)^2 + C$ where $w$ is a scalar parameter (see fig 1(a)). Assuming $w$ is the current value of the parameter, what is the optimal $\eta$ that takes us to the minimum in one step? It is easy to visualize that, as it has been known since Newton, the optimal $\eta$ is the inverse of the second derivative of $E$, i.e. $1/h$. Any smaller or slightly larger value will yield slower convergence. A value more then twice the optimal will cause divergence.

In multidimension, things are more complicated. If the objective function is quadratic, the surfaces of equal cost are ellipsoids (or ellipses in 2D as shown on figure 1(b)). Intuitively, if the learning rate is set for optimal convergence along the direction of largest second derivative, then it will be small enough to ensure (slow) convergence along all the other directions. This corresponds to setting the learning rate to the inverse of the second derivative *in the direction in which it is the largest*. The largest learning rate that ensures convergence is twice that value. The actual optimal $\eta$ is somewhere in between. Setting it to the inverse of the largest second derivative is both safe, and close enough to the optimal. The second derivative information is contained in the *Hessian matrix* of $E(W)$: the symmetric matrix $H$ whose $(i, j)$ component is $\partial^2 E(W)/\partial w_i \partial w_j$. If the learning machine has $N$ free parameters (weights), $H$ is an $N$ by $N$ matrix. The Hessian can be decomposed (diagonalized) into a product of the form $H = R\Lambda R^T$, where $\Lambda$ is a diagonal matrix whose diagonal terms (the *eigenvalues* of $H$) are the second derivatives of $E(W)$

along the principal axes of the ellipsoids of equal cost, and $R$ is a rotation matrix which defines the directions of these principal axes. The direction of largest second derivative is the principal eigenvector of $H$, and the largest second derivative is the corresponding eigenvalue (the largest one). In short, it can be shown that the optimal learning rate is the inverse of the largest eigenvalue of $H$:

$$\eta_{\mathrm{opt}} = \frac{1}{\lambda_{\max}} \qquad (4)$$

## 4   COMPUTING THE HESSIAN'S LARGEST EIGENVALUE WITHOUT COMPUTING THE HESSIAN

This section derives the recipe given in section 2. Large learning machines, such as backpropagation networks can have several thousand free parameters. Computing, or even storing, the full Hessian matrix is often prohibitively expensive. So at first glance, finding its largest eigenvalue in a reasonable time seems rather hopeless. We are about to propose a shortcut based on three simple ideas: 1- the Taylor expansion, 2- the power method, 3- the running average. The method described here is general, and can be applied to any differentiable objective function that can be written as an average over "examples" (e.g. RBFs, or other statistical estimation techniques).

**Taylor expansion:** Although it is often unrealistic to compute the Hessian $H$, there is a simple way to approximate the product of $H$ by a vector of our choosing. Let $\Psi$ be an $N$ dimensional vector, and $\alpha$ a small real constant, the Taylor expansion of the *gradient* of $E(W)$ around $W$ along the direction $\Psi$ gives us

$$H\Psi = \frac{\nabla E(W + \alpha\Psi) - \nabla E(W)}{\alpha} + O(\alpha^2) \qquad (5)$$

Assuming $E$ is locally quadratic (i.e. ignoring the $O(\alpha^2)$ term), the product of $H$ by any vector $\Psi$ can be estimated by subtracting the gradient of $E$ at point $(W + \alpha\Psi)$ from the gradient at $W$. This is an $O(N)$ process, compared to the $O(N^2)$ direct product. In the usual neural network context, this can be done with two forward propagations and two backward propagations. More accurate methods which do not use perturbations for computing $H\Psi$ exist, but they are more complicated to implement than this one. (Pearlmutter, 1993).

**The power method:** Let $\lambda_{\max}$ be the largest eigenvalue[1] of $H$, and $V_{\max}$ the corresponding normalized eigenvector (or a vector in the eigenspace if $\lambda_{\max}$ is degenerate). If we pick a vector $\Psi$ (say, at random) which is non-orthogonal to $V_{\max}$, then iterating the procedure

$$\Psi \leftarrow H\mathcal{N}(\Psi) \qquad (6)$$

will make $\mathcal{N}(\Psi)$ converge to $V_{\max}$, and $\|\Psi\|$ converge to $|\lambda_{\max}|$. The procedure is slow if good accuracy is required, but a good estimate of the eigenvalue can be obtained with a very small number of iterations (typically about 10). The reason for introducing equation (5), is now clear: we can use it to compute the right hand side of (6), yielding

$$\Psi \leftarrow \frac{1}{\alpha} \left( \nabla E\left(W + \alpha\mathcal{N}(\Psi)\right) - \nabla E(W) \right) \qquad (7)$$

where $\Psi$ is the current estimate of the principal eigenvector of $H$, and $\alpha$ is a small constant.

**The "on-line" version:** One iteration of the procedure (7) requires the computation of the gradient of $E$ at two different points of the parameter space. This means that one iteration of (7) is roughly equivalent to two epochs of gradient descent learning (two passes through the entire training set). Since (7) needs to be iterated, say 10 times, the total cost of estimating $\lambda_{max}$ would be approximately equivalent to 20 epochs.

This excessive cost can be drastically reduced with an "on-line" version of (7) which exploits the stationarity of the second-order information over large (and redundant) training sets. Essentially, the hidden "average over patterns" in $\nabla E$ can be replaced by a running average. The procedure becomes

$$\Psi \leftarrow (1-\gamma)\Psi + \gamma\frac{1}{\alpha}\left(\nabla E\left(W + \alpha\mathcal{N}(\Psi)\right) - \nabla E(W)\right) \qquad (8)$$

where $\gamma$ is a small constant which controls the tradeoff between the convergence speed and the accuracy [2]. The "recipe" given in section 2 is a direct implementation of (8). Empirically, this procedure yields sufficiently accurate values in a very short time. In fact, in all the cases we have tried, it converged with only a few dozen *pattern presentations*: a fraction of the time of an entire learning pass through the training set (see the results section). It looks like the essential features of the Hessian can be extracted from only a few examples of the training set. In other words, the largest eigenvalue of the Hessian seems to be mainly determined by the network architecture and initial weights, and by short-term, low-order statistics of the input data. It should be noted that the on-line procedure can only find positive eigenvalues.

## 5    A FEW RESULTS

Experiments will be described for two different network architectures trained on segmented handwritten digits taken from the NIST database. Inputs to the networks were 28x28 pixel images containing a centered and size-normalized image of the character. Network 1 was a 4-hidden layer, locally-connected network with shared weights similar to (Le Cun et al., 1990a) but with fewer feature maps. Each layer is only connected to the layer above. the input is 32x32 (there is a border around the 28x28 image), layer 1 is 2x28x28, with 5x5 convolutional (shared) connections. Layer 2 is 2x14x14 with 2x2 subsampled, averaging connections. Layer 3 is 4x10x10, with 2x5x5 convolutional connections. Layer 4 is 4x5x5 with 2x2 averaging connections, and the output layer is 10x1x1 with 4x5x5 convolutional connections. The network has a total of 64,638 connections but only 1278 free parameters because of the weight sharing. Network 2 was a regular 784x30x10 fully-connected network (23860 weights). The sigmoid function used for all units in both nets was $1.7159\tanh(2/3x)$. Target outputs were set to +1 for the correct unit, and -1 for the others.

To check the validity of our assumptions, we computed the full Hessian of Network 1 on 300 patterns (using finite differences on the gradient) and obtained the eigenvalues and eigenvectors using one of the EISPACK routines. We then computed

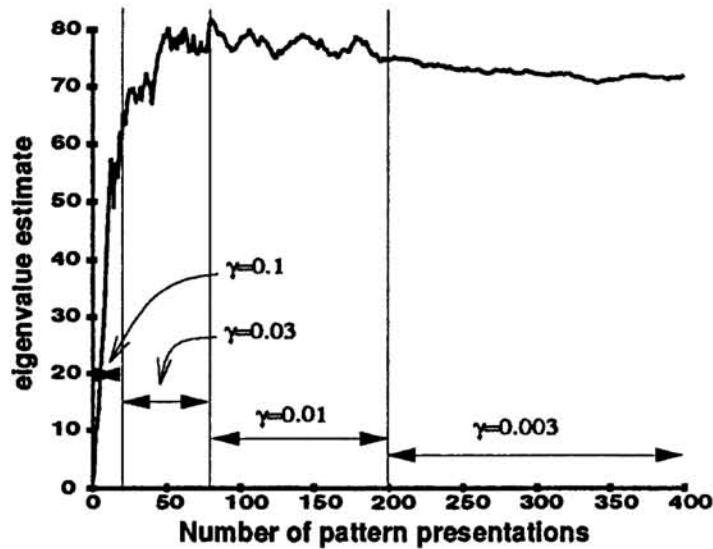

Figure 2: Convergence of the on-line eigenvalue estimation (Network 1)

the principal eigenvector and eigenvalue using procedures (7), and (8). All three methods agreed within less than a percent on the eigenvalue. An example run of (8) on a 1000 pattern set is shown on figure 2. A 10% accurate estimate of the largest eigenvalue is obtained in less than 200 pattern presentations (one fifth of the database). As can be seen, the value is fairly stable over small portions of the set, which means that increasing the set size would not require more iterations of the estimation procedure.

A second series of experiments were run to verify the accuracy of the learning rate prediction. Network 1 was trained on 1000 patterns, and network 2 on 300 patterns, both with SGD. Figure 3 shows the Mean Squared Error of the two networks after 1,2,3,4 and 5 passes through the training set as a function of the learning rate, for one particular initial weight vector. The constant $\gamma$ was set to 0.1 for the first 20 patterns, 0.03 for the next 60, 0.01 for the next 120, and 0.003 for the next 200 (400 total pattern presentations), but it was found that adequate values were obtained after only 100 to 200 pattern presentations. The vertical bar represents the value predicted by the method for that particular run. It is clear that the predicted optimal value is very close to the correct optimal learning rate. Other experiments with different training sets and initial weights gave similar results. Depending on the initial weights, the largest eigenvalue for Network 1 varied between 80 and 250, and for Network 2 between 250 and 400. Experiments tend to suggest that the optimal learning rate varies only slightly during the early phase of training. The learning rate may need to be decreased for long learning sessions, as SGD converts from the "getting near the minimum" mode to the "wobbling around" mode.

There are many other method for adjusting the learning rate. Unfortunately, most of them are based on some measurement of the oscillations of the gradient (Jacobs, 1987). Therefore, they are difficult to apply to stochastic gradient descent.

# 6   MORE ON EIGENVALUES AND EIGENVECTORS

We believe that computing the optimal learning rate is only one of many applications of our eigenvector estimation technique. The procedure can be adapted to serve many applications.

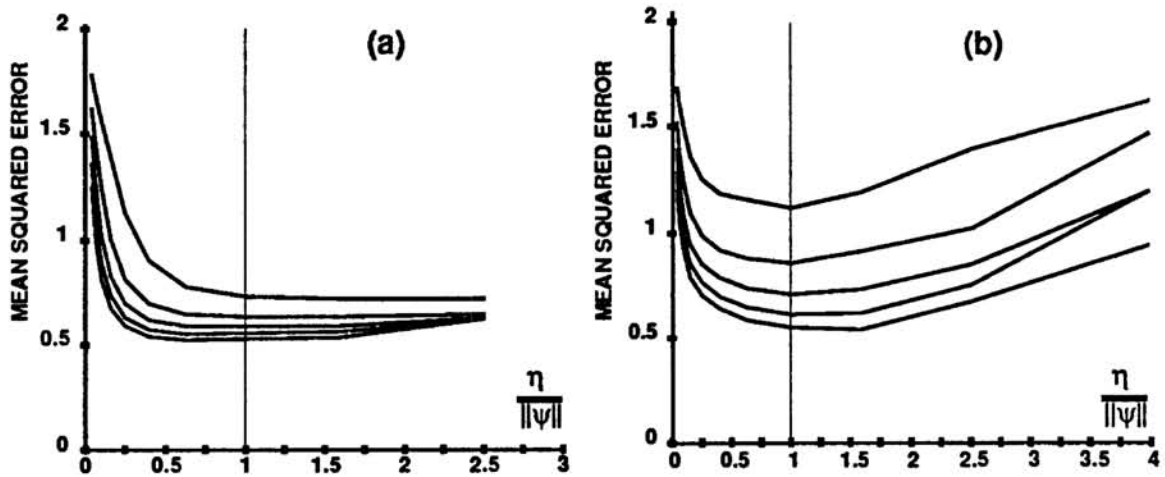

Figure 3: Mean Squared Error after 1,2,3,4, and 5 epochs (from top to bottom) as a function of the ratio between the learning rate $\eta$ and the learning rate predicted by the proposed method $\|\Psi\|^{-1}$. (a) Network 1 trained on 1000 patterns, (b) Network 2 trained on 300 patterns.

An important variation of the learning rate estimation is when, instead of update rule 3, we use a "scaled SGD" rule of the form $W \leftarrow W - \eta\Phi\nabla E^p(W)$, where $\Phi$ is a diagonal matrix (each weight has its own learning rate $\eta\phi_i$). For example, each $\phi_i$ can be the inverse of the corresponding diagonal term of the average Hessian, which can be computed efficiently as suggested in (Le Cun, 1987; Becker and Le Cun, 1988). Then procedure 8 must be changed to

$$\Psi \leftarrow (1-\gamma)\Psi + \gamma\frac{1}{\alpha}\Phi^{\frac{1}{2}}\left(\nabla E\left(W + \alpha\Phi^{\frac{1}{2}}\mathcal{N}(\Psi)\right) - \nabla E(W)\right) \qquad (9)$$

where the terms of $\Phi^{\frac{1}{2}}$ are the square root of the corresponding terms in $\Phi$. More generally, the above formula applies to any transformation of the parameter space whose Jacobian is $\Phi^{\frac{1}{2}}$. The added cost is small since $\Phi^{\frac{1}{2}}$ is diagonal.

Another extension of the procedure can compute the first $K$ principal eigenvectors and eigenvalues. The idea is to store $K$ eigenvector estimates $\Psi_k$, $k = 1 \ldots K$, updated simultaneously with equation (8) (this costs a factor $K$ over estimating only one). We must also ensure that the $\Psi_k$'s remain orthogonal to each other. This can be performed by projecting each $\Psi_k$ onto the space orthogonal to the space subtended by the $\Psi_l$, $l < k$. This is an $NK$ process, which is relatively cheap if the network uses shared weights. A generalization of the acceleration method introduced in (Le Cun, Kanter and Solla, 1991) can be implemented with this technique. The idea is to use a "Newton-like" weight update formula of the type

$$W \leftarrow W - \sum_{k=1}^{K}\|\Psi_k\|^{-1}P_k$$

where $P_k$, $k = 1 \ldots K - 1$ is the projection of $\nabla E(W)$ onto $\Psi_k$, and $P_K$ is the projection of $\nabla E(W)$ on the space orthogonal to the $\Psi_k$, $(k = 1 \ldots K - 1)$. In theory, this procedure can accelerate the training by a factor $\|\Psi_1\|/\|\Psi_K\|$, which is between 3 and 10 for $K = 5$ in a typical backprop network. Results will be reported in a later publication.

Interestingly, the method can be slightly modified to yield the *smallest* eigenvalues/eigenvectors. First, the largest eigenvalue $\lambda_{\max}$ must be computed (or bounded

above). Then, by iterating

$$\Psi \leftarrow (1 - \gamma)\Psi + \lambda_{\max}\mathcal{N}(\Psi) - \gamma\frac{1}{\alpha}\left(\nabla E\left(W + \alpha\mathcal{N}(\Psi)\right) - \nabla E(W)\right) \qquad (10)$$

one can compute the eigenvector corresponding to the smallest (probably negative) eigenvalue of $(H - \lambda_{\max}I)$, which is the same as $H$'s. This can be used to determine the direction(s) of displacement in parameter space that will cause the least increase of the objective function. There are obvious applications of this to weight elimination methods: a better version of OBD (Le Cun et al., 1990b) or a more efficient version of OBS (Hassibi and Stork, 1993).

We have proposed efficient methods for (a) computing the product of the Hessian by any vector, and (b) estimating the few eigenvectors of largest or smallest eigenvalues. The methods were successfully applied the estimation of the optimal learning rate in Stochastic Gradient Descent learning We feel that we have only scratched the surface of the many applications of the proposed techniques.

## Acknowledgements

Yann LeCun and Patrice Simard would like to thank the members of the Adaptive Systems Research dept for their support and comments. Barak Pearlmutter was partially supported by grants NSF ECS-9114333 and ONR N00014-92-J-4062 to John Moody.

## Footnotes

[1] largest in absolute value, not largest algebraically

[2]the procedure (8) is not an unbiased estimator of (7). Large values of $\gamma$ are likely to produce slightly underestimated eigenvalues, but this inaccuracy has no practical consequences.

## References

Becker, S. and Le Cun, Y. (1988). Improving the Convergence of Back-Propagation Learning with Second-Order Methods. Technical Report CRG-TR-88-5, University of Toronto Connectionist Research Group.

Hassibi, B. and Stork, D. (1993). Optimal Brain Surgeon. In Giles, L., Hanson, S., and Cowan, J., editors, *Advances in Neural Information Processing Systems*, volume 5, (Denver, 1992). Morgan Kaufman.

Jacobs, R. A. (1987). Increased Rates of Convergence Through Learning Rate Adaptation. Department of Computer and Information Sciences COINS-TR-87-117, University of Massachusetts, Amherst, Ma.

Le Cun, Y. (1987). *Modeles connexionnistes de l'apprentissage (connectionist learning models)*. PhD thesis, Université P. et M. Curie (Paris 6).

Le Cun, Y., Boser, B., Denker, J. S., Henderson, D., Howard, R. E., Hubbard, W., and Jackel, L. D. (1990a). Handwritten digit recognition with a back-propagation network. In Touretzky, D., editor, *Advances in Neural Information Processing Systems 2 (NIPS*89)*, Denver, CO. Morgan Kaufman.

Le Cun, Y., Denker, J. S., Solla, S., Howard, R. E., and Jackel, L. D. (1990b). Optimal Brain Damage. In Touretzky, D., editor, *Advances in Neural Information Processing Systems 2 (NIPS*89)*, Denver, CO. Morgan Kaufman.

Le Cun, Y., Kanter, I., and Solla, S. (1991). Eigenvalues of covariance matrices: application to neural-network learning. *Physical Review Letters*, 66(18):2396–2399.

Moller, M. (1992). supervised learning on large redundant training sets. In *Neural Networks for Signal Processing 2*. IEEE press.

Pearlmutter, B. (1993). Phd thesis, Carnegie-Mellon University, Pittsburgh PA.

Widrow, B. and Stearns, S. D. (1985). *Adaptive Signal Processing*. Prentice-Hall.